# Forward-Decoding Kernel-Based Phone Sequence Recognition

**Shantanu Chakrabartty** and **Gert Cauwenberghs**
Center for Language and Speech Processing
Department of Electrical and Computer Engineering
Johns Hopkins University, Baltimore MD 21218
{*shantanu,gert*}*@jhu.edu*

## Abstract

Forward decoding kernel machines (FDKM) combine large-margin classifiers with hidden Markov models (HMM) for maximum a posteriori (MAP) adaptive sequence estimation. State transitions in the sequence are conditioned on observed data using a kernel-based probability model trained with a recursive scheme that deals effectively with noisy and partially labeled data. Training over very large datasets is accomplished using a sparse probabilistic support vector machine (SVM) model based on quadratic entropy, and an on-line stochastic steepest descent algorithm. For speaker-independent continuous phone recognition, FDKM trained over 177,080 samples of the TIMIT database achieves 80.6% recognition accuracy over the full test set, without use of a prior phonetic language model.

## 1 Introduction

Sequence estimation is at the core of many problems in pattern recognition, most notably speech and language processing. Recognizing dynamic patterns in sequential data requires a set of tools very different from classifiers trained to recognize static patterns in data assumed *i.i.d.* distributed over time.

The speech recognition community has predominantly relied on hidden Markov models (HMMs) [1] to produce state-of-the-art results. HMMs are generative models that function by estimating probability densities and therefore require a large amount of data to estimate parameters reliably. If the aim is discrimination between classes, then it might be sufficient to model discrimination boundaries between classes which (in most affine cases) afford fewer parameters.

Recurrent neural networks have been used to extend the dynamic modeling power of HMMs with the discriminant nature of neural networks [2], but learning long term dependencies remains a challenging problem [3]. Typically, neural network training algorithms are prone to local optima, and while they work well in many situations, the quality and consistency of the converged solution cannot be warranted.

Large margin classifiers, like support vector machines, have been the subject of intensive research in the neural network and artificial intelligence communities [4]. They are attractive because they generalize well even with relatively few data points in the training set, and bounds on the generalization error can be directly obtained from the training data. Under general conditions, the training procedure finds a unique solution (decision or regression surface) that provides an out-of-sample performance superior to many techniques.

Recently, support vector machines (SVMs) [4] have been used for phoneme (or phone) recognition [5] and have shown encouraging results. However, use of a standard SVM

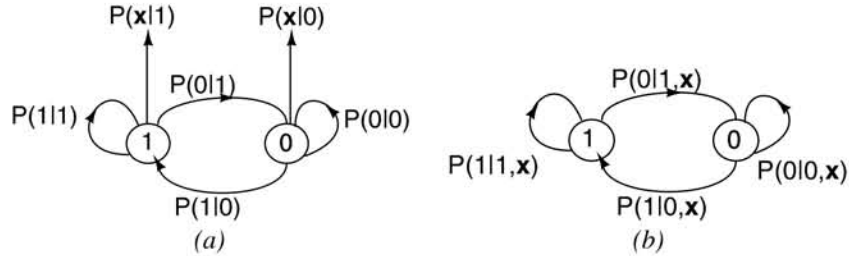

Figure 1: *(a) Two state Markovian maximum-likehood (ML) model with static state transition probabilities and observation vectors* **x** *emitted from the states. (b) Two state Markovian MAP model, where transition probabilities between states are modulated by the observation vector* **x**.

classifier by itself implicitly assumes *i.i.d.* data, unlike the sequential nature of phones.

To model inter-phonetic dependencies, maximum likelihood (ML) approaches assume a phonetic language model that is independent of the utterance data [6], as illustrated in Figure 1 (a). In contrast, the maximum a posteriori (MAP) approach assumes transitions between states that are directly modulated by the observed data, as illustrated in Figure 1 (b). The MAP approach lends itself naturally to hybrid HMM/connectionist approaches with performance comparable to state-of-the-art HMM systems [7].

FDKM [8] can be seen a hybrid HMM/SVM MAP approach to sequence estimation. It thereby augments the ability of large margin classifiers to infer sequential properties of the data. FDKMs have shown superior performance for channel equalization in digital communication where the received symbol sequence is contaminated by inter symbol interference [8].

In the present paper, FDKM is applied to speaker-independent continuous phone recognition. To handle the vast amount of data in the TIMIT corpus, we present a sparse probabilistic model and efficient implementation of the associated FDKM training procedure.

## 2  FDKM formulation

The problem of FDKM recognition is formulated in the framework of MAP (maximum a posteriori) estimation, combining Markovian dynamics with kernel machines. A Markovian model is assumed with symbols belonging to $S$ classes, as illustrated in Figure 1(a) for $S = 2$. Transitions between the classes are modulated in probability by observation (data) vectors **x** over time.

### 2.1  Decoding Formulation

The MAP forward decoder receives the sequence $\overline{\mathbf{X}}[n] = \{\mathbf{x}[n], \mathbf{x}[n-1], \ldots, \mathbf{x}[1]\}$ and produces an estimate of the probability of the state variable $q[n]$ over all classes $i$, $\alpha_i[n] = P(q[n] = i \mid \overline{\mathbf{X}}[n], \mathbf{w})$, where **w** denotes the set of parameters for the learning machine. Unlike *hidden* Markov models, the states directly encode the symbols, and the observations **x** modulate transition probabilities between states [7]. Estimates of the posterior probability $\alpha_i[n]$ are obtained from estimates of local transition probabilities using the *forward-decoding* procedure [7]

$$\alpha_i[n] = \sum_{j=0}^{S-1} P_{ij}[n]\, \alpha_j[n-1] \qquad (1)$$

where $P_{ij}[n] = P(q[n] = i \mid q[n-1] = j, \mathbf{x}[n], \mathbf{w})$ denotes the probability of making a transition from class $j$ at time $n-1$ to class $i$ at time $n$, given the current observation vector $\mathbf{x}[n]$. The forward decoding (1) embeds sequential dependence of the data wherein the probability estimate at time instant $n$ depends on all the previous data. An on-line

estimate of the symbol $q[n]$ is thus obtained:

$$q^{\text{est}}[n] = \arg\max_i \alpha_i[n] \tag{2}$$

The BCJR forward-backward algorithm [9] produces in principle a better estimate that accounts for future context, but requires a backward pass through the data, which is impractical in many applications requiring real time decoding.

Accurate estimation of transition probabilities $P_{ij}[n]$ in (1) is crucial in decoding (2) to provide good performance. In [8] we used kernel logistic regression [10], with regularized maximum cross-entropy, to model conditional probabilities. A different probabilistic model that offers a sparser representation is introduced below.

## 2.2 Training Formulation

For training the MAP forward decoder, we assume access to a training sequence with labels (class memberships). For instance, the TIMIT speech database comes labeled with phonemes. Continuous (soft) labels could be assigned rather than binary indicator labels, to signify uncertainty in the training data over the classes. Like probabilities, label assignments are normalized: $\sum_{i=0}^{S-1} y_i[n] = 1, y_i[n] \geq 0$.

The objective of training is to maximize the cross-entropy of the estimated probabilities $\alpha_i[n]$ given by (1) with respect to the labels $y_i[n]$ over all classes $i$ and training data $n$

$$H = \sum_{n=0}^{N-1} \sum_{i=0}^{S-1} y_i[n] \log \alpha_i[n] \tag{3}$$

To provide capacity control we introduce a regularizer $\Omega(\mathbf{w})$ in the objective function [11]. The parameter space $\mathbf{w}$ can be partitioned into disjoint parameter vectors $\mathbf{w}_{ij}$ and $b_{ij}$ for each pair of classes $i, j = 0, \ldots, S-1$ such that $P_{ij}[n]$ depends only on $\mathbf{w}_{ij}$ and $b_{ij}$. (The parameter $b_{ij}$ corresponds to the bias term in the standard SVM formulation). The regularizer can then be chosen as the $L_2$ norm of each disjoint parameter vector, and the objective function becomes

$$H = C \sum_{n=0}^{N-1} \sum_{i=0}^{S-1} y_i[n] \log \alpha_i[n] - \frac{1}{2} \sum_{j=0}^{S-1} \sum_{i=0}^{S-1} |\mathbf{w}_{ij}|^2 \tag{4}$$

where the regularization parameter $C$ controls complexity versus generalization as a bias-variance trade-off [11]. The objective function (4) is similar to the primal formulation of a large margin classifier [4]. Unlike the convex (quadratic) cost function of SVMs, the formulation (4) does *not* have a unique solution and direct optimization could lead to poor local optima. However, a *lower bound* of the objective function can be formulated so that maximizing this lower bound reduces to a set of convex optimization sub-problems with an elegant dual formulation in terms of support vectors and kernels. Applying the convex property of the $-\log(.)$ function to the convex sum in the forward estimation (1), we obtain directly

$$H \geq \sum_{j=0}^{S-1} H_j \tag{5}$$

where

$$H_j = \sum_{n=0}^{N-1} C_j[n] \sum_{i=0}^{S-1} y_i[n] \log P_{ij}[n] - \frac{1}{2} \sum_{i=0}^{S-1} |\mathbf{w}_{ij}|^2 \tag{6}$$

with effective regularization sequence

$$C_j[n] = C\alpha_j[n-1] . \tag{7}$$

Disregarding the intricate dependence of (7) on the results of (6) which we defer to the following section, the formulation (6) is equivalent to regression of conditional probabilities $P_{ij}[n]$ from labeled data $\mathbf{x}[n]$ and $y_i[n]$, for a given outgoing state $j$.

### 2.3 Kernel Logistic Probability Regression

Estimation of conditional probabilities $\Pr(i|\mathbf{x})$ from training data $\mathbf{x}[n]$ and labels $y_i[n]$ can be obtained using a regularized form of kernel logistic regression [10]. For each outgoing state $j$, one such probabilistic model can be constructed for the incoming state $i$ conditional on $\mathbf{x}[n]$:

$$P_{ij}[n] = \exp(f_{ij}(\mathbf{x}[n]))/\sum_{s=0}^{S-1}\exp(f_{sj}(\mathbf{x}[n])) \tag{8}$$

As with SVMs, dot products in the expression for $f_{ij}(\mathbf{x})$ in (8) convert into kernel expansions over the training data $\mathbf{x}[m]$ by transforming the data to feature space [12]

$$
\begin{aligned}
f_{ij}(\mathbf{x}) \quad &= \quad \mathbf{w}_{ij}.\mathbf{x} + b_{ij} \\
&= \quad \sum_m \lambda_{ij}^m\, \mathbf{x}[m].\mathbf{x} + b_{ij} \\
&\xrightarrow{\Phi(\cdot)} \quad \sum_m \lambda_{ij}^m\, K(\mathbf{x}[m],\mathbf{x}) + b_{ij}
\end{aligned}
\tag{9}
$$

where $K(\cdot,\cdot)$ denotes any symmetric positive-definite kernel[1] that satisfies the Mercer condition, such as a Gaussian radial basis function or a polynomial [11].

Optimization of the lower-bound in (5) requires solving $M$ disjoint but similar suboptimization problems (6). The subscript $j$ is omitted in the remainder of this section for clarity. The (primal) objective function of kernel logistic regression expresses regularized cross-entropy (6) of the logistic model (8) in the form [13, 14]

$$H = -\sum_i \frac{1}{2}|\mathbf{w}_i|^2 + C\sum_m^N[\sum_i^M y_i[m]f_k(\mathbf{x}[m]) - \log(e^{f_1(\mathbf{x}[m])} + ... + e^{f_M(\mathbf{x}[m])})] . \tag{10}$$

The parameters $\lambda_{ij}^m$ in (9) are determined by minimizing a dual formulation of the objective function (10) obtained through the Legendre transformation, which for logistic regression takes the form of an entropy-based potential function in the parameters [10]

$$H_e = \sum_i^M[\frac{1}{2}\sum_l^N\sum_m^N \lambda_i^l Q_{lm}\lambda_i^m + C\sum_m^N(y_i[m] - \lambda_i^m/C)\log(y_i[m] - \lambda_i^m/C)] \tag{11}$$

subject to constraints

$$\sum_m \lambda_i^m \quad = \quad 0 \tag{12}$$

$$\sum_i \lambda_i^m \quad = \quad 0 \tag{13}$$

$$\lambda_i^m \quad \leq \quad Cy_i[m] \tag{14}$$

There are two disadvantages of using the logistic regression dual directly:

1. The solution is non-sparse and all the training points contribute to the final solution. For tasks involving large data sets like phone recognition this turns out to be prohibitive due to memory and run-time constraints.

2. Even though the dual optimization problem is convex, it is not quadratic and precludes the use of standard quadratic programming (QP) techniques. One has to resort to Newton-Raphson or other nonlinear optimization techniques which complicate convergence and require tuning of additional system parameters.

### 2.4 *Gini*SVM formulation

The *Gini*SVM probabilistic model [15] provides a sparse alternative to logistic regression. A quadratic ('Gini' [16]) index replaces entropy in the dual formulation of logistic regression. The 'Gini' index provides a lower bound of the dual logistic functional, and its quadratic form produces sparse solutions as with support vector machines. The tightness of the bound provides an elegant trade-off between approximation and sparsity.

Jensen's inequality ($\log p \leq p - 1$) formulates the lower bound for the entropy term in (11) in the form of the multivariate *Gini* impurity index [16]:

$$1 - \sum_i^M p_i^2 \leq - \sum_i^M p_i \log p_i \qquad (15)$$

where $0 \leq p_i \leq 1, \forall i$ and $\sum_i p_i = 1$. Both forms of entropy $-\sum_i^M p_i \log p_i$ and $1 - \sum_i^M p_i^2$ reach their maxima at the same values $p_i \equiv 1/M$ corresponding to a uniform distribution. As in the binary case, the bound can be tightened by scaling the *Gini* index with a multiplicative factor $\gamma \geq 1$, of which the particular value depends on $M$.[2] The *Gini*SVM dual cost function $H_g$ is then given by

$$H_g = \sum_i^M [\frac{1}{2} \sum_l^N \sum_m^N \lambda_i^l Q_{lm} \lambda_i^m + \gamma C (\sum_m^N (y_i[m] - \lambda_i^m/C)^2 - 1)] \qquad (16)$$

The convex quadratic cost function (16) with constraints in (11) can now be minimized directly using standard quadratic programming techniques. The primary advantage of the technique is that it yields sparse solutions and yet approximates the logistic regression solution very well [15].

### 2.5 Online *Gini*SVM Training

For very large datasets such as TIMIT, using a QP approach to train *Gini*SVM may still be prohibitive even through sparsity drastically in the trained model reduces the number of support vectors. An on-line estimation procedure is presented, that computes each coefficient $\lambda_i^n$ in turn from single presentation of the data $\{\mathbf{x}[n], y_i[n]\}$. A line search in the parameter $\lambda_i^n$ and the bias $b_i$ performs stochastic steepest descent of the dual objective function (16) of the form

$$\lambda_i^n = C\big(y_i[n] - \frac{[Cy_i[n](Q_{nn} + 2) + f_i(x[n]) + 2\sum_\ell^n \lambda_i^\ell - z^n]_+}{C(Q_{nn} + 2) + 2\gamma}\big) \qquad (17)$$

$$b_i \leftarrow b_i + \sum_l^n \lambda_i^l \qquad (18)$$

where $[x]_+$ denotes the positive part of $x$. The normalization factor $z^n$ is determined by equation

$$\sum_i^M [Cy_i[n](Q_{nn} + 2) + f_i[n] + 2\sum_\ell^n \lambda_i^\ell - z^n]_+ = C(Q_{nn} + 2) + 2\gamma \qquad (19)$$

solved in at most $M$ algorithmic iterations.

## 3 Recursive FDKM Training

The weights (7) in (6) are recursively estimated using an iterative procedure reminiscent of (but different from) expectation maximization. The procedure involves computing new estimates of the sequence $\alpha_j[n-1]$ to train (6) based on estimates of $P_{ij}$ using previous values of the parameters $\lambda_{ij}^m$. The training proceeds in a series of epochs, each refining the

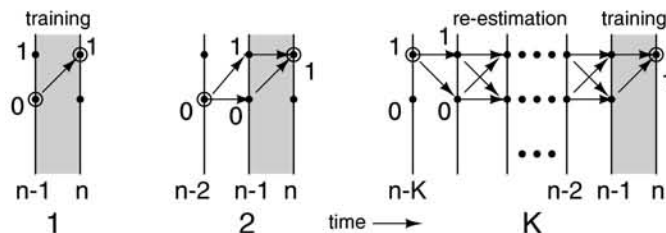

Figure 2: *Iterations involved in training FDKM on a trellis based on the Markov model of Figure 1. During the initial epoch, parameters of the probabilistic model, conditioned on the observed label for the outgoing state at time $n-1$, of the state at time $n$ are trained from observed labels at time $n$. During subsequent epochs, probability estimates of the outgoing state at time $n-1$ over increasing forward decoding depth $k = 1, \ldots K$ determine weights assigned to data $n$ for training each of the probabilistic models conditioned on the outgoing state.*

estimate of the sequence $\alpha_j[n-1]$ by increasing the size of the time window (decoding depth, $k$) over which it is obtained by the forward algorithm (1).

The training steps are illustrated in Figure 2 and summarized as follows:

1. To bootstrap the iteration for the first training epoch ($k = 1$), obtain initial values for $\alpha_j[n-1]$ from the labels of the outgoing state, $\alpha_j[n-1] = y_j[n-1]$. This corresponds to taking the labels $y_i[n-1]$ as true state probabilities which corresponds to the standard procedure of using fragmented data to estimate transition probabilities.

2. Train logistic kernel machines, one for each outgoing class $j$, to estimate the parameters in $P_{ij}[n]$, $i, j = 1, .., S$ from the training data $\mathbf{x}[n]$ and labels $y_i[n]$, weighted by the sequence $\alpha_j[n-1]$.

3. Re-estimate $\alpha_j[n-1]$ using the forward algorithm (1) over increasing decoding depth $k$, by initializing $\alpha_j[n-k]$ to $y[n-k]$.

4. Re-train, increment decoding depth $k$, and re-estimate $\alpha_j[n-1]$, until the final decoding depth is reached ($k = K$).

The performance of FDKM training depends on the final decoding depth $K$, although observed variations in generalization performance for large values of $K$ are relatively small. A suitable value can be chosen *a priori* to match the extent of temporal dependency in the data. For phoneme classification in speech, the decoding depth can be chosen according to the length of a typical syllable.

An efficient procedure to implement the above algorithm is discussed in [15].

## 4  Experiments and Results

The performance of FDKM was evaluated on the full TIMIT dataset [17], consisting of labeled continuous spoken utterances. The 60 phone classes presented in TIMIT were first collapsed onto 39 classes according to standard folding techniques [6]. The training set consisted of 6,300 sentences spoken by 63 speakers, resulting in 177,080 phone instances. The test set consisted of 192 sentences spoken by 24 speakers.

The speech signal was first processed by a pre-emphasis filter with transfer function $1 - 0.97z^{-1}$. Subsequently, a 25 ms Hamming window was applied over 10 ms shifts to extract a sequence of phonetic segments. Cepstral coefficients were extracted from the sequence, combined with their first and second order time differences into a 39-dimensional vector. Cepstral mean subtraction and speaker normalization were subsequently applied.

Table 1: Performance Evaluation of FDKM ($K = 10$) on TIMIT

| Machine | Accuracy | Insertion | Substitution | Deletion | Errors |
|---|---|---|---|---|---|
| FDKM, $C = 0.5$ | 79.9% | 6.3% | 14.7% | 5.4% | 26.4% |
| FDKM, $C = 1$ | 80.6% | 6.3% | 14.3% | 5.1% | 25.7% |
| FDKM, $C = 2.5$ | 79.6% | 5.3% | 14.6% | 5.8% | 25.7% |

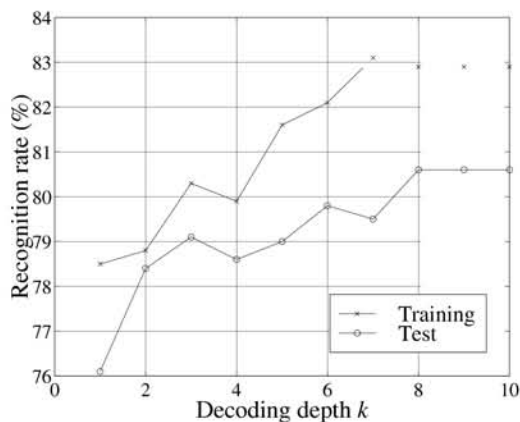

Figure 3: *Recognition rate as a function of decoding depth $k = 1, \ldots K$.*

Each phone utterance were then subdivided into three segments with relative proportions 4:3:4 [18]. The features in the three segments were individually averaged and concatenated to obtain a 117-dimensional feature vector.

Evaluation on the test was performed using thresholding of state probabilities in the MAP forward decoding (2) [19], with threshold 0.25. The decoded phone sequence was then compared with the transcribed sequence using Levenshtein's distance to evaluate different sources of errors. Multiple runs of identical phones in the decoded and transcribed sequences were collapsed to single phone instances to reflect true insertion errors.

Table 1 summarizes the results of the experiments with FDKM on TIMIT for different values of the regularization constant $C$. The recognition performance is comparable to the state of the art using HMMs and other approaches, in the upper 70% and lower 80% range [2, 5, 20]. Figure 3 illustrates the improvement in recognition rate with increasing decoding depth $k$. The optimum value $k \approx 10$ corresponds to inter-phonetic dependencies on a time scale of 100 ms.

## 5 Conclusion

Experiments with FDKM on the TIMIT corpus have demonstrated levels of speaker-independent continuous phone recognition accuracy comparable to or better than other approaches that use HMMs and their various extensions. FDKM improves decoding and generalization performance for data with embedded sequential structure, providing an elegant tradeoff between learning temporal versus spatial dependencies. The recursive estimation procedure reduces or masks the effect of noisy or missing labels $y_j[n]$. Further improvements can be expected by tuning of hyper-parameters and improved representation of acoustic features.

**Acknowledgement**

This work was supported by a grant from the Catalyst Foundation.

## Footnotes

[1] $K(\mathbf{x},\mathbf{y}) = \Phi(\mathbf{x}).\Phi(\mathbf{y})$. The map $\Phi(\cdot)$ need not be computed explicitly, as it only appears in inner-product form.

[2]Unlike the binary case ($M = 2$), the factor $\gamma$ for general $M$ cannot be chosen to match the two maxima at $p_i = 1/M$.

# References

[1] L. Rabiner and B-H Juang, *Fundamentals of Speech Recognition*, Englewood Cliffs, NJ: Prentice-Hall, 1993.

[2] Robinson, A.J., "An application of recurrent nets to phone probability estimation," *IEEE Transactions on Neural Networks*, vol. 5,No.2,March 1994.

[3] Bengio, Y., "Learning long-term dependencies with gradient descent is difficult," *IEEE T. Neural Networks*, vol. 5, pp. 157-166, 1994.

[4] Vapnik, V. *The Nature of Statistical Learning Theory*, New York: Springer-Verlag, 1995.

[5] Clark, P. and Moreno, M.J. "On the use of Support Vector Machines for Phonetic Classification," IEEE Conf. Proc., 1999.

[6] Lee, K.F. and Hon, H.W, "Speaker-Independent phone recognition using hidden markov models," *IEEE Transactions on Acoustics, Speech and Signal Processing*, vol. 37, pp. 1641-1648, 1989.

[7] Bourlard, H. and Morgan, N., *Connectionist Speech Recognition: A Hybrid Approach*, Kluwer Academic, 1994.

[8] Chakrabartty, S. and Cauwenberghs, G. "Sequence Estimation and Channel Equalization using Forward Decoding Kernel Machines," *IEEE Int. Conf. Acoustics and Signal Proc. (ICASSP'2002)*, Orlando FL, 2002.

[9] Bahl, L.R., Cocke J., Jelinek F. and Raviv J. "Optimal decoding of linear codes for minimizing symbol error rate," *IEEE Transactions on Inform. Theory*, vol. **IT-20**, pp. 284-287, 1974.

[10] Jaakkola, T. and Haussler, D. "Probabilistic kernel regression models," *Proceedings of Seventh International Workshop on Artificial Intelligence and Statistics* , 1999.

[11] Girosi, F., Jones, M. and Poggio, T. "Regularization Theory and Neural Networks Architectures," *Neural Computation*, vol. **7**, pp 219-269, 1995.

[12] Schölkopf, B., Burges, C. and Smola, A., Eds., *Advances in Kernel Methods-Support Vector Learning*, MIT Press, Cambridge, 1998.

[13] Wahba, G. *Support Vector Machine, Reproducing Kernel Hilbert Spaces and Randomized GACV*, Technical Report 984, Department of Statistics, University of Wisconsin, Madison WI.

[14] Zhu, J and Hastie, T., "Kernel Logistic Regression and Import Vector Machine," *Adv. IEEE Neural Information Processing Systems (NIPS'2001)*, Cambridge, MA: MIT Press, 2002.

[15] Chakrabartty, S. and Cauwenberghs, G. "Forward Decoding Kernel Machines: A hybrid HMM/SVM Approach to Sequence Recognition," *IEEE Int. Conf. of Pattern Recognition: SVM workshop. (ICPR'2002)*, Niagara Falls, 2002.

[16] Breiman, L. Friedman, J. H. et al. *Classification and Regression Trees*, Wadsworth and Brooks, Pacific Grove, CA, 1984.

[17] Fisher, W., Doddington G. et al *The DARPA Speech Recognition Research Database: Specifications and Status*. Proceedings DARPA speech recognition workshop, pp. 93-99, 1986.

[18] Fosler-Lussier, E. Greenberg, S. Morgan, N., "Incorporating contextual phonetics into automatic speech recognition," *Proc. XIVth Int. Cong. Phon. Sci.*, 1999.

[19] Wald, A. *Sequential Analysis*, Wiley, New York, 1947.

[20] Chengalvarayan, R. and Deng, Li., "Speech Trajectory Discrimination Using the Minimum Classification Error Training, " *IEEE Transactions on Speech and Audio Processing*, vol. 6, pp. 505-515, Nov. 1998.
